# Learning Multiple Related Tasks using Latent Independent Component Analysis

**Jian Zhang†,   Zoubin Ghahramani†‡,   Yiming Yang†**

| | |
|---|---|
| † School of Computer Science | ‡ Gatsby Computational Neuroscience Unit |
| Cargenie Mellon University | University College London |
| Pittsburgh, PA 15213 | London WC1N 3AR, UK |

{jian.zhang, zoubin, yiming}@cs.cmu.edu

## Abstract

We propose a probabilistic model based on Independent Component Analysis for learning multiple related tasks. In our model the task parameters are assumed to be generated from independent sources which account for the relatedness of the tasks. We use Laplace distributions to model hidden sources which makes it possible to identify the hidden, independent components instead of just modeling correlations. Furthermore, our model enjoys a sparsity property which makes it both parsimonious and robust. We also propose efficient algorithms for both empirical Bayes method and point estimation. Our experimental results on two multi-label text classification data sets show that the proposed approach is promising.

## 1   Introduction

An important problem in machine learning is how to generalize between multiple related tasks. This problem has been called "multi-task learning", "learning to learn", or in some cases "predicting multivariate responses". Multi-task learning has many potential practical applications. For example, given a newswire story, predicting its subject categories as well as the regional categories of reported events based on the same input text is such a problem. Given the mass tandem spectra of a sample protein mixture, identifying the individual proteins as well as the contained peptides is another example.

Much attention in machine learning research has been placed on how to effectively learn multiple tasks, and many approaches have been proposed[1][2][3][4][5][6][10][11]. Existing approaches share the basic assumption that tasks are related to each other. Under this general assumption, it would be beneficial to learn all tasks jointly and borrow information from each other rather than learn each task independently. Previous approaches can be roughly summarized based on how the "relatedness" among tasks is modeled, such as IID tasks[2], a Bayesian prior over tasks[2][6][11], linear mixing factors[5][10], rotation plus shrinkage[3] and structured regularization in kernel methods[4].

Like previous approaches, the basic assumption in this paper is that the multiple tasks are related to each other. Consider the case where there are $K$ tasks and each task is a binary

classification problem from the same input space (e.g., multiple simultaneous classifications of text documents). If we were to separately learn a classifier, with parameters $\theta_k$ for each task $k$, we would be ignoring relevant information from the other classifiers. The assumption that the tasks are related suggests that the $\theta_k$ for different tasks should be related to each other. It is therefore natural to consider different statistical models for how the $\theta_k$'s might be related.

We propose a model for multi-task learning based on Independent Component Analysis (ICA)[9]. In this model, the parameters $\theta_k$ for different classifiers are assumed to have been generated from a sparse linear combination of a small set of basic classifiers. Both the coefficients of the sparse combination (the factors or sources) and the basic classifiers are learned from the data. In the multi-task learning context, the relatedness of multiple tasks can be explained by the fact that they share certain number of hidden, independent components. By controlling the model complexity in terms of those independent components we are able to achieve better generalization capability. Furthermore, by using distributions like Laplace we are able to enjoy a sparsity property, which makes the model both parsimonious and robust in terms of identifying the connections with independent sources. Our model can be combined with many popular classifiers, and as an indispensable part we present scalable algorithms for both empirical Bayes method and point estimation, with the later being able to solve high-dimensional tasks. Finally, being a probabilistic model it is always convenient to obtain probabilistic scores and confidence which are very helpful in making statistical decisions. Further discussions on related work are given in Section 5.

## 2   Latent Independent Component Analysis

The model we propose for solving multiple related tasks, namely the Latent Independent Component Analysis (LICA) model, is a hierarchical Bayesian model based on the traditional Independent Component Analysis. ICA[9] is a promising technique from signal processing and designed to solve the blind source separation problem, whose goal is to extract independent sources given only observed data that are linear combinations of the unknown sources. ICA has been successfully applied to blind source separation problem and shows great potential in that area. With the help of non-Gaussianity and higher-order statistics it can correctly identify the independent sources, as opposed to technique like Factor Analysis which is only able to remove the correlation in the data due to the intrinsic Gaussian assumption in the corresponding model.

In order to learn multiple related tasks more effectively, we transform the joint learning problem into learning a generative probabilistic model for our tasks (or more precisely, task parameters), which precisely explains the relatedness of multiple tasks through the latent, independent components. Unlike the standard Independent Component Analysis where we use observed data to estimate the hidden sources, in LICA the "observed data" for ICA are actually task parameters. Consequently, they are latent and themselves need to be learned from the training data of each individual task. Below we give the precise definition of the probabilistic model for LICA.

Suppose we use $\theta_1, \theta_2, \ldots, \theta_K$ to represent the model parameters of $K$ tasks where $\theta_k \in \mathbb{R}^{F \times 1}$ can be thought as the parameter vector of the $k$-th individual task. Consider the following generative model for the $K$ tasks:

$$
\begin{aligned}
\theta_k &= \Lambda \mathbf{s}_k + \mathbf{e}_k \\
\mathbf{s}_k &\sim p(\mathbf{s}_k \mid \Phi) \\
\mathbf{e}_k &\sim \mathcal{N}(\mathbf{0}, \Psi)
\end{aligned}
\tag{1}
$$

where $\mathbf{s}_k \in \mathbb{R}^{H \times 1}$ are the hidden source models with $\Phi$ denotes its distribution parameters; $\Lambda \in \mathbb{R}^{F \times H}$ is a linear transformation matrix; and the noise vector $\mathbf{e}_k \in \mathbb{R}^{F \times 1}$

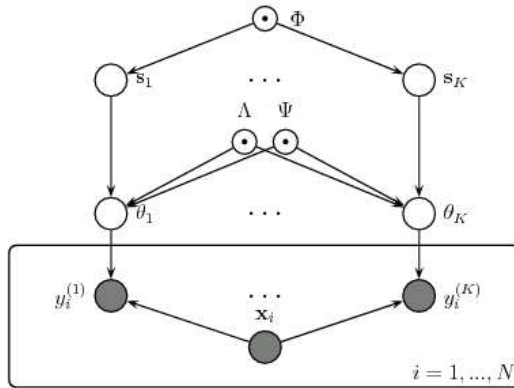

Figure 1: Graphical Model for Latent Independent Component Analysis

is usually assumed to be a multivariate Gaussian with diagonal covariance matrix $\Psi = diag(\psi_{11}, \ldots, \psi_{FF})$ or even $\Psi = \sigma^2 \mathbf{I}$. This is essentially assuming that the hidden sources $\mathbf{s}$ are responsible for all the dependencies among $\theta_k$'s, and conditioned on them all $\theta_k$'s are independent. Generally speaking we can use any member of the exponential families as $p(\mathbf{e}_k)$, but in most situations the noise is taken to be a multivariate Gaussian which is convenient. The graphical model for equation (1) is shown as the upper level in Figure 1, whose lower part will be described in the following.

## 2.1 Probabilistic Discriminative Classifiers

One building block in the LICA is the probabilistic model for learning each individual task, and in this paper we focus on classification tasks. We will use the following notation to describe a probabilistic discriminative classifier for task $k$, and for notation simplicity we omit the task index $k$ below. Suppose we have training data $\mathcal{D} = \{(\mathbf{x}_1, y_1), \ldots, (\mathbf{x}_N, y_N)\}$ where $\mathbf{x}_i \in \mathbb{R}^{F \times 1}$ is the input data vector and $y_i \in \{0, 1\}$ is the binary class label, our goal is to seek a probabilistic classifier whose prediction is based on the conditional probability $p(y = 1 | \mathbf{x}) \stackrel{\triangle}{=} f(\mathbf{x}) \in [0, 1]$. We further assume that the discriminative function to have a linear form $f(\mathbf{x}) = \mu(\theta^T \mathbf{x})$, which can be easily generalized to non-linear functions by some feature mapping. The output class label $y$ can be thought as randomly generated from a Bernoulli distribution with parameter $\mu(\theta^T \mathbf{x})$, and the overall model can be summarized as follows:

$$
\begin{aligned}
y_i &\sim \mathcal{B}(\mu(\theta^T \mathbf{x}_i)) \\
\mu(t) &= \int_{-\infty}^{t} p(z) dz
\end{aligned}
\tag{2}
$$

where $\mathcal{B}(.)$ denotes the Bernoulli distribution and $p(z)$ is the probability density function of some random variable $Z$. By changing the definition of random variable $Z$ we are able to specialize the above model into a variety of popular learning methods. For example, when $p(z)$ is standard logistic distribution we will get logistic regression classifier; when $p(z)$ is standard Gaussian we get the probit regression. In principle any member belonging to the above class of classifiers can be plugged in our LICA, or even generative classifiers like Naive Bayes. We take logistic regression as the basic classifier, and this choice should not affect the main point in this paper. Also note that it is straightforward to extend the framework for regression tasks whose likelihood function $y_i \sim \mathcal{N}(\theta^T \mathbf{x}_i, \sigma^2)$ can be solved by simple and efficient algorithms. Finally we would like to point out that although shown in the graphical model that all training instances share the same input vector $\mathbf{x}$, this is mainly for notation simplicity and there is indeed no such restriction in our model. This is convenient since in reality we may not be able to obtain all the task responses for the same training instance.

# 3 Learning and Inference for LICA

The basic idea of the inference algorithm for the LICA is to iteratively estimate the task parameters $\theta_k$, hidden sources $\mathbf{s}_k$, and the mixing matrix $\Lambda$ and noise covariance $\Psi$. Here we present two algorithms, one for the empirical Bayes method, and the other for point estimation which is more suitable for high-dimensional tasks.

## 3.1 Empirical Bayes Method

The graphical model shown in Figure 1 is an example of a hierarchical Bayesian model, where the upper levels of the hierarchy model the relation between the tasks. We can use an empirical Bayes approach and learn the parameters $\Omega = \{\Phi, \Lambda, \Psi\}$ from the data while treating the variables $\mathcal{Z} = \{\theta_k, \mathbf{s}_k\}_{k=1}^{K}$ as hidden, random variables. To get around the unidentifiability caused by the interaction between $\Lambda$ and $\mathbf{s}$ we assume $\Phi$ is of standard parametric form (e.g. zero mean and unit variance) and thus remove it from $\Omega$. The goal is to learn point estimators $\hat{\Lambda}$ and $\hat{\Psi}$ as well as obtain posterior distributions over hidden variables given training data.

The log-likelihood of incomplete data $\log p(\mathcal{D} \mid \Omega)$ [1] can be calculated by integrating out hidden variables

$$\log p(\mathcal{D}|\Omega) = \sum_{k=1}^{K} \log \left\{ \int \prod_{i=1}^{N} p(y_i^{(k)} \mid \mathbf{x}_i, \theta_k) \left( \int p(\theta_k \mid \mathbf{s}_k, \Lambda, \Psi) p(\mathbf{s}_k|\Phi) d\mathbf{s}_k \right) d\theta_k \right\}$$

for which the maximization over parameters $\Omega = \{\Lambda, \Psi\}$ involves two complicated integrals over $\theta_k$ and $\mathbf{s}_k$, respectively. Furthermore, for classification tasks the likelihood function $p(y|\mathbf{x}, \theta)$ is typically non-exponential and thus exact calculation becomes intractable. However, we can approximate the solution by applying the EM algorithm to decouple it into a series of simpler E-steps and M-steps as follows:

1. E-step: Given the parameter $\Omega^{t-1} = \{\Lambda, \Psi\}^{t-1}$ from the $(t-1)$-th step, compute the distribution of hidden variables given $\Omega^{t-1}$ and $\mathcal{D}$: $p(\mathcal{Z} \mid \Omega^{t-1}, \mathcal{D})$

2. M-step: Maximizing the expected log-likelihood of complete data $(\mathcal{Z}, \mathcal{D})$, where the expectation is taken over the distribution of hidden variables obtained in the E-step: $\Omega^t = \arg\max_\Omega \mathbb{E}_{p(\mathcal{Z}|\Omega^{t-1}, \mathcal{D})} \left[ \log p(\mathcal{D}, \mathcal{Z} \mid \Omega) \right]$

The log-likelihood of complete data can be written as

$$\log p(\mathcal{D}, \mathcal{Z} \mid \Omega) = \sum_{k=1}^{K} \left\{ \sum_{i=1}^{N} \log p(y_i^{(k)} \mid \mathbf{x}_i, \theta_k) + \log p(\theta_k \mid \mathbf{s}_k, \Lambda, \Psi) + \log p(\mathbf{s}_k \mid \Phi) \right\}$$

where the first and third item do not depend on $\Omega$. After some simplification the M-step can be summarized as $\{\hat{\Lambda}, \hat{\Psi}\} = \arg\max_{\Lambda, \Psi} \sum_{k=1}^{K} \mathbb{E}[\log p(\theta_k \mid \mathbf{s}_k, \Lambda, \Psi)]$ which leads to the following updating equations:

$$\hat{\Lambda} = \left( \sum_{k=1}^{K} \mathbb{E}[\theta_k \mathbf{s}_k^T] \right) \left( \sum_{k=1}^{K} \mathbb{E}[\mathbf{s}_k \mathbf{s}_k^T] \right)^{-1} \; ; \;\; \hat{\Psi} = \frac{1}{K} \left( \sum_{k=1}^{K} \mathbb{E}[\theta_k \theta_k^T] - (\sum_{k=1}^{K} \mathbb{E}[\theta_k \mathbf{s}_k^T]) \hat{\Lambda}^T \right)$$

In the E-step we need to calculate the posterior distribution $p(\mathcal{Z} \mid \mathcal{D}, \Omega)$ given the parameter $\Omega$ calculated in previous M-step. Essentially only the first and second order

**Algorithm 1** Variational Bayes for the E-step (subscript $k$ is removed for simplicity)

1. Initialize $q(\mathbf{s})$ with some standard distribution (Laplace distribution in our case): $q(\mathbf{s}) = \prod_{h=1}^{H} \mathcal{L}(0,1)$.

2. Solve the following Bayesian logistic regression (or other Bayesian classifier):

$$q(\theta) \leftarrow \arg\max_{q(\theta)} \left\{ \int q(\theta) \log \frac{\mathcal{N}(\theta; \Lambda \mathbb{E}[\mathbf{s}], \Psi) \prod_{i=1}^{N} p(y_i|\theta, \mathbf{x}_i)}{q(\theta)} d\theta \right\}$$

3. Update $q(\mathbf{s})$:

$$q(\mathbf{s}) \leftarrow \arg\max_{q(\mathbf{s})} \left\{ \int q(\mathbf{s}) \left[ \log \frac{p(\mathbf{s})}{q(\mathbf{s})} - \frac{1}{2}\mathbf{Tr}\left( \Psi^{-1}(\mathbb{E}[\theta\theta^T] + \Lambda\mathbf{s}\mathbf{s}^T\Lambda^T - 2\mathbb{E}[\theta](\Lambda\mathbf{s})^T)\right) \right] d\mathbf{s} \right\}$$

4. Repeat steps 2-5 until convergence conditions are satisfied.

---

moments are needed, namely: $\mathbb{E}[\theta_k]$, $\mathbb{E}[\mathbf{s}_k]$, $\mathbb{E}[\theta_k\theta_k^T]$, $\mathbb{E}[\mathbf{s}_k\mathbf{s}_k^T]$ and $\mathbb{E}[\theta_k\mathbf{s}_k^T]$. Since exact calculation is intractable we will approximate $p(\mathcal{Z} \mid \mathcal{D}, \Omega)$ with $q(\mathcal{Z})$ belonging to the exponential family such that certain distance measure (can be asymmetric) between $p(\mathcal{Z}|\mathcal{D}, \Omega)$ and $q(\mathcal{Z})$) is minimized. In our case we apply the variational Bayes method which applies $\mathcal{KL}\left(q(\mathcal{Z})||p(\mathcal{D}, \mathcal{Z} \mid \Omega)\right)$ as the distance measure. The central idea is to lower bound the log-likelihood using Jensen's inequality: $\log p(\mathcal{D}) = \log \int p(\mathcal{D}, \mathcal{Z})d\mathcal{Z} \geq \int q(\mathcal{Z}) \log \frac{p(\mathcal{D}, \mathcal{Z})}{q(\mathcal{Z})}d\mathcal{Z}$. The RHS of the above equation is what we want to maximize, and it is straightforward to show that maximizing this lower bound is equivalent to minimize the KL-divergence $\mathcal{KL}(q(\mathcal{Z})||p(\mathcal{Z}|\mathcal{D}))$. Since given $\Omega$ the $K$ tasks are decoupled, we can conduct inference for each task respectively. We further assume $q(\theta_k, \mathbf{s}_k) = q(\theta_k)q(\mathbf{s}_k)$, which in general is a reasonable simplifying assumption and allows us to do the optimization iteratively. The details for the E-step are shown in Algorithm 1.

We would like to comment on several things in Algorithm 1. First, we assume the form of $q(\theta)$ to be multivariate Gaussian, which is a reasonable choice especially considering the fact that only the first and second moments are needed in the M-step. Second, the prior choice of $p(\mathbf{s})$ in step 3 is significant since for each $\mathbf{s}$ we only have one associated "data point" $\theta$. In particular using the Laplace distribution will lead to a more sparse solution of $\mathbb{E}[\mathbf{s}]$, and this will be made more clear in Section 3.2. Finally, we take the parametric form of $q(\mathbf{s})$ to be the product of Laplace distributions with unit variance but known mean, where the fixed variance is intended to remove the unidentifiability issue caused by the interaction between scales of $\mathbf{s}$ and $\Lambda$. Although using a full covariance Gaussian for $q(\mathbf{s})$ is another choice, again due to unidentifiability reason caused by rotations of $\mathbf{s}$ and $\Lambda$ we could make it a diagonal Gaussian. As a result, we argue that the product of Laplaces is better than the product of Gaussians since it has the same parametric form as the prior $p(\mathbf{s})$.

### 3.1.1 Variational Method for Bayesian Logistic Regression

We present an efficient algorithm based on the variational method proposed in[7] to solve step 2 in Algorithm 1, which is guaranteed to converge and known to be efficient for this problem. Given a Gaussian prior $\mathcal{N}(\mathbf{m}_0, \mathbf{V}_0)$ over the parameter $\theta$ and a training set [2] $\mathcal{D} = \{(\mathbf{x}_1, y_1), \ldots, (\mathbf{x}_N, y_N)\}$, we want to obtain an approximation $\mathcal{N}(\mathbf{m}, \mathbf{V})$ to the true posterior distribution $p(\theta|\mathcal{D})$. Taking one data point $(\mathbf{x}, y)$ as an example, the basic idea is to use an exponential function to approximate the non-exponential likelihood function $p(y|\mathbf{x}, \theta) = (1 + \exp(-y\theta^T\mathbf{x}))^{-1}$ which in turn makes the Bayes formula tractable.

By using the inequality $p(y|\mathbf{x}, \theta) \geq g(\xi) \exp\left\{(y\mathbf{x}^T\theta - \xi)/2 - \lambda(\xi)((\mathbf{x}^T\theta)^2 - \xi^2)\right\} \triangleq p(y|\mathbf{x}, \theta, \xi)$ where $g(z) = 1/(1 + \exp(-z))$ is the logistic function and $\lambda(\xi) = tanh(\xi/2)/4\xi$, we can maximize the lower bound of $p(y|\mathbf{x}) = \int p(\theta)p(y|\mathbf{x}, \theta)d\theta \geq \int p(\theta)p(y|\mathbf{x}, \theta, \xi)d\theta$. An EM algorithm can be formulated by treating $\xi$ as the parameter and $\theta$ as the hidden variable:

- E-step: $\mathbf{Q}(\xi, \xi^t) = \mathbb{E}\left[\log\{p(\theta)p(y|\mathbf{x}, \theta, \xi)\} \mid \mathbf{x}, y, \xi^t\right]$
- M-step: $\xi^{t+1} = \arg\max_\xi \mathbf{Q}(\xi, \xi^t)$

Due to the Gaussianity assumption the E-step can be thought as updating the sufficient statistics (mean and covariance) of $q(\theta)$. Finally by using the Woodbury formula the EM iterations can be unraveled and we get the efficient one-shot E-step updating without involving matrix inversion (due to space limitation we skip the derivation):

$$\mathbf{V}_{post} = \mathbf{V} - \frac{2\lambda(\xi)}{1 + 2\lambda(\xi)c}\mathbf{V}\mathbf{x}(\mathbf{V}\mathbf{x})^T$$

$$\mathbf{m}_{post} = \mathbf{m} - \frac{2\lambda(\xi)}{1 + 2\lambda(\xi)c}\mathbf{V}\mathbf{x}\mathbf{x}^T\mathbf{m} + \frac{y}{2}\mathbf{V}\mathbf{x} - \frac{y}{2}\frac{2\lambda(\xi)}{1 + 2\lambda(\xi)c}c\mathbf{V}\mathbf{x}$$

where $c = \mathbf{x}^T\mathbf{V}\mathbf{x}$, and $\xi$ is calculated first from the M-step which is reduced to find the fixed point of the following one-dimensional problem and can be solved efficiently:

$$\xi^2 = c - \frac{2\lambda(\xi)}{1 + 2\lambda(\xi)c}c^2 + \left(\mathbf{x}^T\mathbf{m} - \frac{2\lambda(\xi)}{1 + 2\lambda(\xi)c}c\mathbf{x}^T\mathbf{m} + \frac{y}{2}c - \frac{y}{2}\frac{2\lambda(\xi)}{1 + 2\lambda(\xi)c}c^2\right)^2$$

And this process will be performed for each data point to get the final approximation $q(\theta)$.

## 3.2   Point Estimation

Although the empirical Bayes method is efficient for medium-sized problem, both its computational cost and memory requirement grow as the number of data instances or features increases. For example, it can easily happen in text or image domain where the number of features can be more than ten thousand, so we need faster methods. We can obtain the point estimation of $\{\theta_k, \mathbf{s}_k\}_{k=1}^K$, by treating it as a limiting case of the previous algorithm. To be more specific, by letting $q(\theta)$ and $q(\mathbf{s})$ converging to the Dirac delta function, step 2 in Algorithm 1 can thought as finding the MAP estimation of $\theta$ and step 4 becomes the following lasso-like optimization problem ($\mathbf{m_s}$ denotes the point estimation of $\mathbf{s}$):

$$\hat{\mathbf{m}}_\mathbf{s} = \arg\min_{\mathbf{m_s}}\left\{2||\mathbf{m_s}||_1 + \mathbf{m}_\mathbf{s}^T\Lambda^T\Psi^{-1}\Lambda\mathbf{m}_\mathbf{s} - 2\mathbf{m}_\mathbf{s}^T\Lambda^T\Psi^{-1}\mathbb{E}[\theta]\right\}$$

which can be solved numerically. Furthermore, the solution of the above optimization is sparse in $\mathbf{m_s}$. This is a particularly nice property since we would only like to consider hidden sources for which the association with tasks are significantly supported by evidence.

## 4   Experimental Results

The LICA model will work most effectively if the tasks we want to learn are very related. In our experiments we apply the LICA model to multi-label text classification problems, which are the case for many existing text collections including the most popular ones like Reuters-21578 and the new RCV1 corpus. Here each individual task is to classify a given document to a particular category, and it is assumed that the multi-label property implies that some of the tasks are related through some latent sources (semantic topics).

For Reuters-21578 we choose nine categories out of ninety categories, which is based on fact that those categories are often correlated by previous studies[8]. After some preprocessing[3] we get 3,358 unique features/words, and empirical Bayes method is used to

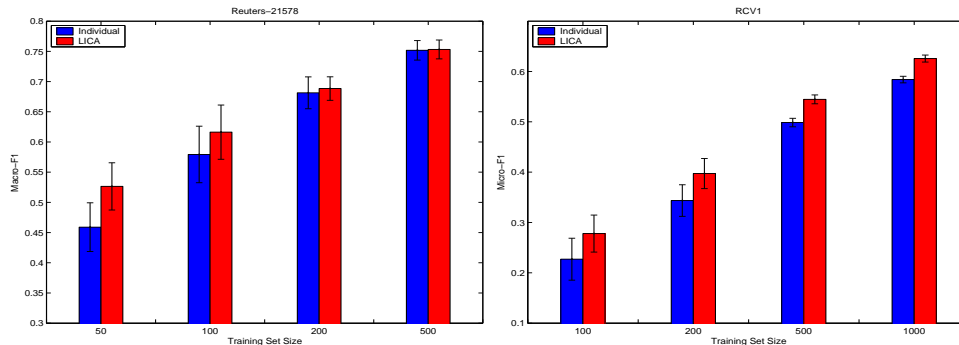

Figure 2: Multi-label Text Classification Results on Reuters-21578 and RCV1

solve this problem. On the other hand, if we include all the 116 TOPIC categories in RCV1 corpus we get a much larger vocabulary size: 47,236 unique features. Bayesian inference is intractable for this high-dimensional case since memory requirement itself is $O(F^2)$ to store the full covariance matrix $\mathbb{V}[\theta]$. As a result we take the point estimation approach which reduces the memory requirement to $O(F)$. For both data sets we use the standard training/test split, but for RCV1 since the test part of corpus is huge (around 800k documents) we only randomly sample 10k as our test set. Since the effectiveness of learning multiple related tasks jointly should be best demonstrated when we have limited resources, we evaluate our LICA by varying the size of training set. Each setting is repeated ten times and the results are summarized in Figure 2.

In Figure 2 the result "individual" is obtained by using regularized logistic regression for each category individually. The number of tasks $K$ is equal to 9 and 116 for the Reuters-21578 and the RCV1 respectively, and we set $H$ (the dimension of hidden source) to be the same as $K$ in our experiments. We use F1 measure which is preferred to error rate in text classification due to the very unbalanced positive/negative document ratio. For the Reuters-21578 collection we report the Macro-F1 results because this corpus is easier and thus Micro-F1 are almost the same for both methods. For the RCV1 collection we only report Micro-F1 due to space limitation, and in fact we observed similar trend in Macro-F1 although values are much lower due to the large number of rare categories. Furthermore, we achieved a sparse solution for the point estimation method. In particular, we obtained less than 5 non-zero sources out of 116 for most of the tasks for the RCV1 collection.

## 5 Discussions on Related Work

By viewing multitask learning as predicting multivariate responses, Breiman and Friedman[3] proposed a method called "Curds and Whey" for regression problems. The intuition is to apply shrinkage in a rotated basis instead of the original task basis so that information can be borrowed among tasks.

By treating tasks as IID generated from some probability space, empirical process theory[2] has been applied to study the bounds and asymptotics of multiple task learning, similar to the case of standard learning. On the other hand, from the general Bayesian perspective[2][6] we could treat the problem of learning multiple tasks as learning a Bayesian prior over the task space. Despite the generality of above two principles, it is often necessary to assume some specific structure or parametric form of the task space since the functional space is usually of higher or infinite dimension compared to the input space.

Our model is related to the recently proposed Semiparametric Latent Factor Model (SLFM) for regression by Teh et. al.[10]. It uses Gaussian Processes (GP) to model regression through a latent factor analysis. Besides the difference between FA and ICA, its advantage

is that GP is non-parametric and works on the instance space; the disadvantage of that model is that training instances need to be shared for all tasks. Furthermore, it is not clear how to explore different task structures in this instance-space viewpoint. As pointed out earlier, the exploration of different source models is important in learning related tasks as the prior often plays a more important role than it does in standard learning.

## 6   Conclusion and Future Work

In this paper we proposed a probabilistic framework for learning multiple related tasks, which tries to identify the shared latent independent components that are responsible for the relatedness among those tasks. We also presented the corresponding empirical Bayes method as well as point estimation algorithms for learning the model. Using non-Gaussian distributions for hidden sources makes it possible to identify independent components instead of just decorrelation, and in particular we enjoyed the sparsity by modeling hidden sources with Laplace distribution. Having the sparsity property makes the model not only parsimonious but also more robust since the dependence on latent, independent sources will be shrunk toward zero unless significantly supported by evidence from the data. By learning those related tasks jointly, we are able to get a better estimation of the latent independent sources and thus achieve a better generalization capability compared to conventional approaches where the learning of each task is done independently. Our experimental results in multi-label text classification problems show evidence to support our claim.

Our approach assumes that the underlying structure in the task space is a linear subspace, which can usually capture important information about independent sources. However, it is possible to achieve better results if we can incorporate specific domain knowledge about the relatedness of those tasks into the model and obtain a reliable estimation of the structure. For future research, we would like to consider more flexible source models as well as incorporate domain specific knowledge to specify and learn the underlying structure.

## Footnotes

[1] Here with a little abuse of notation we ignore the difference of discriminative and generative at the classifier level and use $p(\mathcal{D} \mid \theta_k)$ to denote the likelihood in general.

[2]Again we omit the task index $k$ and use $y \in \{-1, 1\}$ instead of $y \in \{0, 1\}$ to simplify notation.

[3]We do stemming, remove stopwords and rare words (words that occur less than three times).

## References

[1] Ando, R. and Zhang, T. A Framework for Learning Predicative Structures from Multiple Tasks and Unlabeled Data. Technical Rerport RC23462, IBM T.J. Watson Research Center, 2004.

[2] Baxter, J. A Model for Inductive Bias Learning. *J. of Artificial Intelligence Research*, 2000.

[3] Breiman, L. and Friedman J. Predicting Multivariate Responses in Multiple Linear Regression. J. Royal Stat. Society B, 59:3-37, 1997.

[4] Evgeniou, T., Micchelli, C. and Pontil, M. Learning Multiple Tasks with Kernel Methods. *J. of Machine Learning Research*, 6:615-637, 2005.

[5] Ghosn, J. and Bengio, Y. Bias Learning, Knowledge Sharing. *IEEE Transaction on Neural Networks*, 14(4):748-765, 2003.

[6] Heskes, T. Empirical Bayes for Learning to Learn. In *Proc. of the 17th ICML*, 2000.

[7] Jaakkola, T. and Jordan, M. A Variational Approach to Bayesian Logistic Regression Models and Their Extensions. In *Proc. of the Sixth Int. Workshop on AI and Statistics*, 1997.

[8] Koller, D. and Sahami, M. Hierarchically Classifying Documents using Very Few Words. In *Proc. of the 14th ICML*, 1997.

[9] Roberts, S. and Everson, R. (editors). *Independent Component Analysis: Principles and Practice*, Cambridge University Press, 2001.

[10] Teh, Y.-W., Seeger, M. and Jordan, M. Semiparametric Latent Factor Models. In Z. Ghahramani and R. Cowell, editors, *Workshop on Artificial Intelligence and Statistics 10*, 2005.

[11] Yu, K., Tresp, V. and Schwaighofer, A. Learning Gaussian Processes from Multiple Tasks. In *Proc. of the 22nd ICML*, 2005.
